# How to Combine Expert (or Novice) Advice when Actions Impact the Environment

**Daniela Pucci de Farias**[*]
Department of Mechanical Engineering
Massachusetts Institute of Technology
Cambridge, MA 02139
pucci@mit.edu

**Nimrod Megiddo**
IBM Almaden Research Center
650 Harry Road, K53-B2
San Jose, CA 95120
megiddo@almaden.ibm.com

## Abstract

The so-called "experts algorithms" constitute a methodology for choosing actions repeatedly, when the rewards depend both on the choice of action and on the unknown current state of the environment. An experts algorithm has access to a set of strategies ("experts"), each of which may recommend which action to choose. The algorithm learns how to combine the recommendations of individual experts so that, in the long run, for any fixed sequence of states of the environment, it does as well as the best expert would have done relative to the same sequence. This methodology may not be suitable for situations where the evolution of states of the environment depends on past chosen actions, as is usually the case, for example, in a repeated non-zero-sum game.

A new experts algorithm is presented and analyzed in the context of repeated games. It is shown that asymptotically, under certain conditions, it performs as well as the best available expert. This algorithm is quite different from previously proposed experts algorithms. It represents a shift from the paradigms of regret minimization and myopic optimization to consideration of the long-term effect of a player's actions on the opponent's actions or the environment. The importance of this shift is demonstrated by the fact that this algorithm is capable of inducing cooperation in the repeated Prisoner's Dilemma game, whereas previous experts algorithms converge to the suboptimal non-cooperative play.

## 1 Introduction

**Experts algorithms.** A well-known class of methods in machine learning are the so-called *experts algorithms*. The goal of these methods is to learn from experience how to combine advice from multiple experts in order to make sequential decisions in an online environment. The general idea can be described as follows. An agent has to choose repeatedly from a given set of actions. The reward in each stage is a function of the chosen action and the choices of Nature or the environment (also referred to as the "adversary" or the "opponent"). A set of strategies $\{1, \ldots, r\}$ is available for the agent to choose from. We refer

---

[*]Work done while at IBM Almaden Research Center, San Jose, California.

to each such strategy as an "expert," even though some of them might be simple enough to be called a "novice." Each expert suggests a choice of an action based on the history of the process and the expert's own choice algorithm. After each stage, the agent observes his own reward. An experts algorithm directs the agent with regard to which expert to follow in the next stage, based on the past history of actions and rewards.

**Minimum Regret.** A popular criterion in decision processes is called Minimum Regret (MR). Regret is defined as the difference between the reward that could have been achieved, given the choices of Nature, and what was actually achieved. An expert selection rule is said to minimize regret if it yields an average reward as large as that of any single expert, against any fixed sequence of actions chosen by the opponent. Indeed, certain experts algorithms, which at each stage choose an expert from a probability distribution that is related to the reward accumulated by the expert prior to that stage, have been shown to minimize regret [1, 2]. It is crucial to note though that, since the experts are compared on a sequence-by-sequence basis, the MR criterion ignores the possibility that different experts may induce different sequences of choices by the opponent. Thus, MR makes sense only under the assumption that Nature's choices are independent of the decision maker's choices.

**Repeated games.** We consider a multi-agent interaction in the form of a repeated game. In repeated games, the assumption that the opponent's choices are independent of the agent's choices is not justified, because the opponent is likely to base his choices of actions on the past history of the game. This is evident in nonzero-sum games, where players are faced with issues such as how to coordinate actions, establish trust or induce cooperation. These goals require that they take each other's past actions into account when making decisions. But even in the case of zero-sum games, the possibility that an opponent has bounded rationality may lead a player to look for patterns to be exploited in the opponent's past actions.

We illustrate some of the aforementioned issues with an example involving the Prisoner's Dilemma game.

**The Prisoner's Dilemma.** In the single-stage Prisoner's Dilemma (PD) game, each player can either cooperate (C) or defect (D). Defecting is better than cooperating regardless of what the opponent does, but it is better for both players if both cooperate than if both defect. Consider the repeated PD. Suppose the row player consults with a set of experts, including the "defecting expert," who recommends defection all the time. Let the strategy of the column player in the repeated game be fixed. In particular, the column player may be very patient and cooperative, willing to wait for the row player to become cooperative, but eventually becoming non-cooperative if the row player does not seem to cooperate. Since defection is a dominant strategy in the stage game, the defecting expert achieves in each step a reward as high as any other expert against any sequence of choices of the column player, so the row player learns with the experts algorithm to defect all the time. Obviously, in retrospect, this seems to minimize regret, since for any fixed sequence of actions by the column player, constant defection is the best response. Obviously, constant defection is not the best response in the repeated game against many possible strategies of the column player. For instance, the row player would regret very much using the experts algorithm if he were told later that the column player had been playing a strategy such as Tit-for-Tat.[1]

In this paper, we propose and analyze a new experts algorithm, which follows experts judiciously, attempting to maximize the long-term average reward. Our algorithm differs from previous approaches in at least two ways. First, each time an expert is selected, it is followed for multiple stages of the game rather than a single one. Second, our algorithm takes

into account only the rewards that were actually achieved by an expert in the stages it was followed, rather than the reward that could have been obtained in any stage. Our algorithm enjoys the appealing simplicity of the previous algorithms, yet it leads to a qualitatively different behavior and improved average reward. We present two results:

1. A "worst-case" guarantee that, in any play of the game, our algorithm achieves an average reward that is asymptotically as large as that of the expert that did best in the rounds of the game when it was played. The worst-case guarantee holds without any assumptions on the opponent's or experts' strategies.

2. Under certain conditions, our algorithm achieves an average reward that is asymptotically as large as the average reward that could have been achieved by the best expert, had it been followed exclusively. The conditions are required in order to facilitate learning and for the notion of a "best expert" to be well-defined.

The effectiveness of the algorithm is demonstrated by its performance in the repeated PD game, namely, it is capable of identifying the opponent's willingness to cooperate and it induces cooperative behavior.

The paper is organized as follows. The algorithm is described in section 2. A bound based on actual expert performance is presented in section 3. In section 4, we introduce and discuss an assumption about the opponent. This assumption gives rise to asymptotic optimality, which is presented in section 5.

## 2   The algorithm

We consider an "experts strategy" for the row player in a repeated two-person game in normal form. At each stage of the game, the row and column player choose actions $i \in I$ and $j \in J$, respectively. The row player has a reward matrix $R$, with entries $0 \leq R_{ij} \leq u$. The row player may consult at each stage with a set of experts $\{1, \ldots, r\}$, before choosing an action for the next stage. We denote by $\sigma_e$ the strategy proposed by expert $e$, i.e., $\sigma_e = \sigma_e(h_s)$ is the proposed probability distribution over actions in stage $s$, given the history $h_s$. We refer to the row player as the agent and to the column player as the opponent.

Usually, the form of experts algorithms found in the literature is as follows. Denote by $M_e(s - 1)$ the average reward achieved by expert $e$ prior to stage $s$ of the game[2]. Then, a reasonable rule is to follow expert $e$ in stage $s$ with a probability that is proportional to some monotone function of $M_e(s - 1)$. In particular, when this probability is proportional to $\exp\{\eta_s M_e(s-1)\}$, for a certain choice of $\eta_s$, this algorithm is known to minimize regret [1, 2]. Specifically, by letting $j_s$ ($s = 1, 2, \ldots$) denote the observed actions of the opponent up to stage $s$, and letting $\sigma_X$ denote the strategy induced by the experts algorithm, we have

$$\frac{1}{s}\sum_{s'=1}^{s} \mathrm{E}[R(i, j_s) : i \sim \sigma_X(h_s)] \geq \sup_e \frac{1}{s}\sum_{s'=1}^{s} \mathrm{E}[R(i, j_s) : i \sim \sigma_e(h_s)] - o(s). \quad (1)$$

The main deficiency of the regret minimization approach is that it fails to consider the influence of chosen actions of a player on the future choices of the opponent — the inequality (1) holds for any *fixed* sequence $(j_s)$ of the opponent's moves, but does not account for the fact that different choices of actions by the agent may induce different sequences of the opponent. This subtlety is also missing in the experts algorithm we described above. At each

stage of the game, the selection of expert is based solely on how well various experts have, or could have, done so far. There is no notion of learning how an expert's actions affect the opponent's moves. For instance, in the repeated PD game described in the introduction, assuming that the opponent is playing Tit-for-Tat, the algorithm is unable to establish the connection between the opponent's cooperative moves and his own.

Based on the previous observations, we propose a new experts algorithm, which takes into account how the opponent reacts to each of the experts. The idea is simple: instead of choosing a (potentially different) expert at each stage of the game, the number of stages an expert is followed, each time it is selected, increases gradually. We refer to each such set of stages as a "phase" of the algorithm. Following is the statement of the *Strategic Experts Algorithm* (SEA). The phase number is denoted by $i$. The number of phases during which expert $e$ has been followed is denoted by $N_e$. The average payoff from phases in which expert $e$ has been followed is denoted by $M_e$.

**Strategic Experts Algorithm (SEA):**

1. For $e = 1, \ldots, r$, set $M_e = N_e = 0$. Set $i = 1$.

2. With probability $1/i$ perform an *exploration phase*, namely, choose an expert $e$ from the uniform distribution over $\{1, \ldots, r\}$; otherwise, perform an *exploitation phase*, namely, choose an expert $e$ from the uniform distribution over the set of experts $e'$ with maximum $M_{e'}$.

3. Set $N_e = N_e + 1$. Follow expert $e$'s instructions for the next $N_e$ stages. Denote by $\tilde{R}$ the average payoff accumulated during the current phase (i.e., these $N_e$ stages), and set
$$M_e = M_e + \tfrac{2}{N_e+1} (\tilde{R} - M_e) \ .$$

4. Set $i = i + 1$ and go to step 2.

Throughout the paper, $s$ will denote a stage number, and $i$ will denote a phase number. We denote by $M_1(i), \ldots, M_r(i)$ the values of the registers $M_1, \ldots, M_r$, respectively, at the end of phase $i$. Similarly, we denote by $N_1(i), \ldots, N_r(i)$ the values of the registers $N_1, \ldots, N_r$, respectively, at the end of phase $i$. Thus, $M_e(i)$ and $N_e(i)$ are, respectively, the average payoff accumulated by expert $e$ and the total number of phases this expert was followed on or before phase $i$. We will also let $M(s)$ and $M(i)$ denote, without confusion, the average payoff accumulated by the algorithm in the first $s$ stages or first $i$ phases of the game.

## 3   A bound based on actual expert performance

When the SEA is employed, the average reward $M_e(i)$ that was actually achieved by each available expert $e$ is being tracked. It is therefore interesting to compare the average reward $M(s)$ achieved by the SEA, with the averages achieved by the various experts. The following theorem states that, in the long run, the SEA obtains almost surely at least as much as the actual average reward obtained by any available expert during the same play.

**Theorem 3.1.**
$$\Pr \left( \liminf_{s \to \infty} M(s) \geq \max_e \liminf_{i \to \infty} M_e(i) \right) = 1 \ . \tag{2}$$

Although the claim of Theorem 3.1 seems very close to regret minimization, there is an essential difference in that we compare the average reward of our algorithm with the average reward *actually achieved* by each expert in the stages when it was played, as opposed to the estimated average reward based on the whole history of play of the opponent.

Note that the bound (2) is merely a statement about the average reward of the SEA in comparison to the average reward achieved by each expert, but nothing is claimed about the limits themselves. Theorem 5.1 proposes an application of this bound in a case when an additional assumption about the experts' and opponent's strategies allows us to analyze convergence of the average reward for each expert. Another interesting case occurs when one of the experts plays a maximin strategy; in this case, bound (2) ensures that the SEA achieves at least the maximin value of the game. The same holds if one of the experts is a regret-minimizing experts algorithm, which is known to achieve at least the maximin value of the game.

The remainder of this section consists of a sketch of the proof of Theorem 3.1.

**Sketch of proof:** Denote by $V$ be the random variable $\max_e \liminf_{i \to \infty} M_e(i)$, and denote by $\bar{E}$ the expert that achieves that maximum (if there is more than one, let $\bar{E}$ be the one with the least index). For any logical proposition $L$, let $\delta(L) = 1$ if $L$ is true; otherwise $\delta(L) = 0$. The proof of Theorem 3.1 relies on establishing that, for all $\epsilon > 0$ and any expert $e$,

$$\Pr\left(\lim_{i \to \infty} \frac{N_e(i) \cdot \delta(M_e(i) \leq V - \epsilon)}{i} = 0\right) = 1 \,. \tag{3}$$

In words, if the average reward of an expert falls below $V$ by a non negligible amount, it must have been followed only a small fraction of the total number of phases. There are three possible situations for any expert $e$: (a) When $\liminf_{i \to \infty} M_e(i) > V - \epsilon$, the inequality is satisfied trivially. (b) When $\limsup_{i \to \infty} M_e(i) < V$, there is a phase $I$ such that for all $i \geq I$, $M_e(i) < M_{\bar{E}}(i)$, so that expert $e$ is played only on exploration phases, and a large deviations argument establishes that (3) holds. (c) The most involved situation occurs when $\liminf_{i \to \infty} M_e(i) \leq V - \epsilon$ and $\limsup_{i \to \infty} M_e(i) \geq V$. To show that (3) holds in this case, we are going to focus on the trajectory of $M_e(i)$ each time it goes from above $V - \epsilon/2$ to below $V - \epsilon + \delta/2$, for some $0 < \delta < \epsilon$. We offer the two following observations:

1. Let $I_k$ be the $k^{th}$ phase such that $M_e(i) \leq V - \epsilon + \delta/2$, and let $I_k^0$ be the first phase before $I_k$ such that $M_e(i) \geq V - \epsilon/2$. Then, between phases $I_k^0$ and $I_k$, expert $e$ is selected at least $N_e(I_k^0)(\epsilon - \delta)/(6u)$ times.
   Denoting by $I_k^j, j = 1, \ldots, P_k$, the phases when expert $e$ is selected, between $I_k^0$ and $I_k$, we have

   $$M_e(I_k^j) \geq \frac{M_e(I_k^{j-1})(N_e(I_k^0) + j - 1)(N_e(I_k^0) + j)}{(N_e(I_k^0) + j)(N_e(I_k^0) + j + 1)} \,.$$

   A simple induction argument shows that, in order to have

   $$M_e(I_k) \leq V - \frac{\epsilon}{2} \leq M_e(I_k^0) - \frac{\epsilon - \delta}{2} \,,$$

   expert $e$ must be selected a number of times $P_k \geq N_e(I_k^0)(\epsilon - \delta)/(6u)$.

2. For all large enough $k$, the phases $I_k^j$ when expert $e$ is selected are exclusively exploration phases.
   This follows trivially from the fact that, after a certain phase $I$, we have $M_{\bar{E}}(i) \geq V - \epsilon/2$, for all $i \geq I$, whereas $M_e(i) < V - \epsilon/2$ for all $i$ between $I_k^0$ and $I_k$.

From the first observation, we have

$$\frac{N_e(I_k)}{I_k} \leq \frac{N_e(I_k^0) + P_k}{I_k - I_k^0} \leq \frac{(1 + 6u)P_k}{(\epsilon - \delta)I_k - I_k^0} \,,$$

Since expert $e$ is selected only during exploration phases between $I_k^0$ and $I_k$, a large deviations argument allows us to conclude that the ratio of the number of times $P_k$ expert $e$ is selected, to the total number of phases $I_k - I_k^0$, converges to zero with probability one. We conclude that (3) holds.

We now observe that

$$M(i) = \frac{\sum_e N_e(i)(N_e(i) + 1)M_e(i)}{\sum_e N_e(i)(N_e(i) + 1)}. \tag{4}$$

By a simple optimization argument, we can show that

$$\sum_e N_e(i)(N_e(i) + 1) \geq i(i/r + 1). \tag{5}$$

Using (3) and (5) to bound (4), we conclude that (2) holds for the subsequence of stages $s$ corresponding to the end of each phase of the SEA. It is easy to show that the average reward $M(s)$ in stages $s$ in the middle of phase $i$ becomes arbitrarily close to the average reward at the end of that phase $M(i)$, as $i$ goes to infinity, and the theorem follows . $\quad\square$

## 4   The flexible opponent

In general, it is impossible for an experts algorithm to guarantee, against an unknown opponent, a reward close to what the best available expert would have achieved if it had been the only expert. It is easy to construct examples which prove this impossibility.

**Example: Repeated Matching Pennies.**   In the Matching Pennies (MP) game, each of the player and the adversary has to choose either $H$ ("Heads") or $T$ ("Tails"). If the choices match, the player loses 1; otherwise, he wins 1. A possible strategy for the adversary in the repeated MP game is:

**Adversary:**   Fix a positive integer $s$ and a string $\sigma^s \in \{H, T\}^s$. In each of the first $s$ stages, play the 50:50 mixed strategy. In each of the stages $s+1, s+2, \ldots$, if the sequence of choices of the player during the first $s$ stages coincided with the string $\sigma^s$, then play $T$; otherwise, play the 50:50 mixed strategy.

Suppose each available expert $e$ corresponds to a strategy of the form:

**Expert:**   Fix a string $\sigma_e \in \{H, T\}^s$. During the first $s$ stages play according to $\sigma_e$. In each of the stages $s+1, s+2, \ldots$, play $H$.

Suppose an expert $e^*$ with $\sigma_{e^*} = \sigma^s$ is available. Then, in order for an experts algorithm to achieve at least the reward of $e^*$, it needs to follow the string $\sigma^s$ precisely during the first $s$ stages. Of course, without knowing what $\sigma^s$ is, the algorithm cannot play it with probability one, nor can it learn anything about it during the play.

In view of the repeated MP example, some assumption about the opponent must be made in order for the player to be able to learn how to play to against that opponent. The essence of the difficulty with the above strategy of the opponent is that it is not flexible — the player has only one chance to guess who the best expert is and thus cannot recover from a mistake. Here, we introduce the assumption of *flexibility* as a possible remedy to that problem. Under the assumption of flexibility, the SEA achieves an average reward that is asymptotically as high as what the best expert could be expected to achieve.

**Definition 4.1 (Flexibility).**   *(i) An opponent playing strategy $\pi(s)$ is said to be* flexible *with respect to expert $e$ ($e = 1, \ldots, r$) if there exist constants $\mu_e$, $\tau > 0.25$ and $c$ such that for every stage $s_0$, every possible history $h_{s_0}$ at stage $s_0$ and any number of stages $s$,*

$$\mathbf{E}\left[ \left| \tfrac{1}{s} \sum_{s=s_0+1}^{s_0+s} R(a_e(s), b(s)) - \mu_e \right| \ : \ a_e(s) \sim \sigma_e(h_s), \ b(s) \sim \pi(h_s) \right] \leq \frac{c}{s^\tau}$$

*(ii) Flexibility with respect to a set of experts is defined as flexibility with respect to every member of the set.*

In words, the expected average reward during the $s$ stages between stage $s_0$ and stage $s_0 + s$ converges (as $s$ tends to infinity) to a limit that does not depend on the history of the play prior to stage $s_0$.

**Example 4.1 : Finite Automata.** In the literature on "bounded rationality", players are often modelled as finite automata. A *probabilistic automaton strategy* (PAS) is specified by a tuple $\mathcal{A} = \langle M, O, A, \sigma, P \rangle$, where $M = \{1, \ldots, m\}$ is the finite set of internal states of the automaton, $A$ is the set of possible actions, $\mathcal{O}$ is the set of possible outcomes, $\sigma_i(a)$ is the probability of choosing action $a$ while in state $i$ ($i = 1, \ldots, m$) and $P^o = (P^o_{ij})$ ($1 \leq i, j \leq m$) is the matrix of state transition probabilities, given an outcome $o \in \mathcal{O}$. Thus, at any stage of the game, the automaton picks an action from a probability distribution associated with its current state and transitions into a new state, according to a probability distribution which depends on the outcome of the stage game. If both the opponent and an expert play PASs, then a Markov chain is induced over the set of pairs of the respective internal states. If this Markov chain has a single class of recurrent states, then the flexibility assumption holds. Note that we do not limit the size of the automata; a larger set of internal states implies a slower convergence of the average rewards, but does not affect the asymptotic results for the SEA.

**Example 4.2 : Bounded dependence on the history.** The number of possible histories at stage $s$ grows exponentially with $s$. Thus, it is reasonable to assume that the choice of action would be based not on the exact detail of the history but rather on the empirical distribution of past actions or patterns of actions. If the opponent is believed not to be stationary, then discounting previous observations by recency may be sensible. For instance, if the frequency of play of action $j$ by the opponent is relevant, the player might condition his choice at stage $s + 1$ on the quantities $\tau_j = \sum_{s'=1}^{s} \beta^{s-s'} \delta_{jj_s}$ where $\beta < 1$ and $\delta$ is the Kronecker delta. In this case, only actions $j_s$ at stages $s$ that are relatively recent have a significant impact on $\tau_j$. Therefore strategies based on $\tau_j$ should exhibit behavior similar to that of bounded recall, and lead to flexibility in the same circumstances as the latter.

## 5   A bound based on expected expert performance

In this section we show that if the opponent is "flexible" with respect to the available experts, then the SEA achieves almost surely an average payoff that is asymptotically as large as what the best expert could achieve against the same opponent.

**Theorem 5.1.** *If an opponent $\pi$ is flexible with respect to the experts $1, \ldots, r$, then the average payoff up to stage $s$, $M(s)$, satisfies*

$$\Pr\left(\liminf_{s \to \infty} M(s) \geq \max_e \mu_e\right) = 1 \,.$$

Theorem 5.1 follows from Lemma 5.1, stated and proven below, and Theorem 3.1.

Flexibility comes into play as a way of ensuring that the value of following any given expert is well-defined, and can eventually be estimated as long as the SEA follows that expert sufficiently many times. In other words, flexibility ensures that there is a best expert to be learned, and that learning can effectively occur because actions taken by other experts, which could affect the behavior of the opponent, are eventually forgotten by the latter.

We now present Lemma 5.1, which shows that, under the flexibility assumption, the average reward achieved by each expert is asymptotically almost surely the same as the reward that would have been achieved by the same expert, had he been the only available expert.

**Lemma 5.1.** *If the opponent is flexible with respect to expert $e$, then with probability one, $\lim_{i\to\infty} M_e(i) = \mu_e$.*

**Sketch of proof:** Let $e$ be any expert. By the Borel-Cantelli lemma, exploration occurs infinitely many times, hence $e$ is followed during infinitely many phases. Let $I_j = I_j(e)$, $(j = 1, 2, \ldots)$ be the phase numbers in which $e$ is followed. By Markov's inequality, for every $\epsilon > 0$,

$$\Pr(|M_e(I_j) - \mu_e| > \epsilon) \leq \epsilon^{-4}\mathbf{E}[(M_e(I_j) - \mu_e)^4] \,.$$

If we could show that

$$\sum_{j=1}^{\infty} \mathbf{E}[(M_e(I_j) - \mu_e)^4] < \infty \,, \tag{6}$$

then we could conclude, by the Borel-Cantelli lemma, that with probability one, the inequality $|M_e(I_j) - \mu_e| > \epsilon$ holds only for finitely many values of $j$. This implies that, with probability one, $\lim_{i\to\infty} M_e(i) = \mu_e$. It follows that if the opponent is flexible with respect to expert $e$, then for some $\nu > 0$, as $j$ tends to infinity, $\mathbf{E}[(M_e(I_j) - \mu_e)^4] = O(j^{-1-\nu})$, which suffices for (6). $\qquad\square$

**Example 5.1 : Repeated Prisoner's Dilemma revisited.** Consider playing the repeated PD game against an opponent who plays Tit-for-Tat, and suppose there are only two experts: "Always defect" (AD) and "Always cooperate" (AC). Thus, AC induces cooperation in every stage and yields a payoff higher than AD, which induces defection in every stage of the game except the first one. It is easy to verify that Tit-for-Tat is flexible with respect to the experts AC and AD. Therefore, Theorem 5.1 holds and the SEA achieves an average payoff at least as much as that of AC. By contrast, as mentioned in the introduction, in order to minimize regret, the standard experts algorithm must play D in almost every stage of the game, and therefore achieves a lower payoff.

## Footnotes

[1]The Tit-for-Tat strategy is to play C in the first stage, and later play in every stage whatever the opponent played in the preceding stage.

[2]In different variants of the algorithm and depending on what information is available to the row player, $M_e(s - 1)$ could be either an estimate of the average reward based on reward achieved by expert $e$ in the stages it was played, or the reward it could have obtained, had it been played in all stages against the same history of play of the opponent.

# References

[1] Auer, P., Cesa-Bianchi, N., Freund, Y. & Schapire, R.E. (1995) Gambling in a rigged casino: The adversarial multi-armed bandit problem. In *Proc. 36th Annual IEEE Symp. on Foundations of Computer Science*, pp. 322–331, Los Alamitos, CA: IEEE Computer Society Press.

[2] Freund, Y. & Schapire, R.E. (1999) Adaptive game playing using multiplicative weights. *Games and Economic Behavior* **29**:79–103.

[3] Foster, D. & Vohra, R. (1999) Regret and the on-line decision problem. *Games and Economic Behavior* **29**:7–35.

[4] Fudenberg, D. & Levine, D.K. (1997) *The Theory of Learning in Games.* Cambridge, MA: The MIT Press.

[5] Littlestone, N. & Warmuth, M.K. (1994) The weighted majority algorithm. *Information and Computation* **108** (2):212–261.
